# A PAC-Bayes Risk Bound for General Loss Functions

**Pascal Germain**
Département IFT-GLO
Université Laval
Québec, Canada
Pascal.Germain.1@ulaval.ca

**Alexandre Lacasse**
Département IFT-GLO
Université Laval
Québec, Canada
Alexandre.Lacasse@ift.ulaval.ca

**François Laviolette**
Département IFT-GLO
Université Laval
Québec, Canada
Francois.Laviolette@ift.ulaval.ca

**Mario Marchand**
Département IFT-GLO
Université Laval
Québec, Canada
Mario.Marchand@ift.ulaval.ca

## Abstract

We provide a PAC-Bayesian bound for the expected loss of convex combinations of classifiers under a wide class of loss functions (which includes the exponential loss and the logistic loss). Our numerical experiments with Adaboost indicate that the proposed upper bound, computed on the training set, behaves very similarly as the true loss estimated on the testing set.

## 1 Intoduction

The PAC-Bayes approach [1, 2, 3, 4, 5] has been very effective at providing tight risk bounds for large-margin classifiers such as the SVM [4, 6]. Within this approach, we consider a *prior distribution P* over a space of classifiers that characterizes our prior belief about good classifiers (before the observation of the data) and a *posterior distribution Q* (over the same space of classifiers) that takes into account the additional information provided by the training data. A remarkable result that came out from this line of research, known as the "PAC-Bayes theorem", provides a tight upper bound on the risk of a stochastic classifier (defined on the posterior $Q$) called the *Gibbs classifier*. In the context of binary classification, the $Q$-weighted *majority vote classifier* (related to this stochastic classifier) labels any input instance with the label output by the stochastic classifier with probability more than half. Since at least half of the $Q$ measure of the classifiers err on an example incorrectly classified by the majority vote, it follows that the error rate of the majority vote is at most twice the error rate of the Gibbs classifier. Therefore, given enough training data, the PAC-Bayes theorem will give a small risk bound on the majority vote classifier only when the risk of the Gibbs classifier is small. While the Gibbs classifiers related to the large-margin SVM classifiers have indeed a low risk [6, 4], this is clearly not the case for the majority vote classifiers produced by bagging [7] and boosting [8] where the risk of the associated Gibbs classifier is normally close to 1/2. Consequently, the PAC-Bayes theorem is currently not able to recognize the predictive power of the majority vote in these circumstances.

In an attempt to progress towards a theory giving small risk bounds for low-risk majority votes having a large risk for the associated Gibbs classifier, we provide here a risk bound for convex combinations of classifiers under quite arbitrary loss functions, including those normally used for boosting (like the exponential loss) and those that can give a tighter upper bound to the zero-one loss of weighted majority vote classifiers (like the sigmoid loss). Our numerical experiments with Adaboost [8] indicate that the proposed upper bound for the exponential loss and the sigmoid loss, computed on the training set, behaves very similarly as the true loss estimated on the testing set.

## 2   Basic Definitions and Motivation

We consider binary classification problems where the input space $\mathcal{X}$ consists of an arbitrary subset of $\mathbb{R}^n$ and the output space $\mathcal{Y} = \{-1, +1\}$. An *example* is an input-output $(\mathbf{x}, y)$ pair where $\mathbf{x} \in \mathcal{X}$ and $y \in \mathcal{Y}$. Throughout the paper, we adopt the PAC setting where each example $(\mathbf{x}, y)$ is drawn according to a fixed, but unknown, probability distribution $D$ on $\mathcal{X} \times \mathcal{Y}$. We consider learning algorithms that work in a fixed hypothesis space $\mathcal{H}$ of binary classifiers and produce a *convex combination* $f_Q$ of binary classifiers taken from $\mathcal{H}$. Each binary classifier $h \in \mathcal{H}$ contribute to $f_Q$ with a weight $Q(h) \geq 0$. For any input example $\mathbf{x} \in \mathcal{X}$, the real-valued output $f_Q(\mathbf{x})$ is given by

$$ f_Q(\mathbf{x}) \;=\; \sum_{h \in \mathcal{H}} Q(h) h(\mathbf{x}) \,, $$

where $h(\mathbf{x}) \in \{-1, +1\}$, $f_Q(\mathbf{x}) \in [-1, +1]$, and $\sum_{h \in \mathcal{H}} Q(h) = 1$. Consequently, $Q(h)$ will be called the *posterior* distribution[1].

Since $f_Q(\mathbf{x})$ is also the expected class label returned by a binary classifier randomly chosen according to $Q$, the *margin* $y f_Q(\mathbf{x})$ of $f_Q$ on example $(\mathbf{x}, y)$ is related to the fraction $W_Q(\mathbf{x}, y)$ of binary classifiers that err on $(\mathbf{x}, y)$ under measure $Q$ as follows. Let $I(a) = 1$ when predicate $a$ is true and $I(a) = 0$ otherwise. We then have:

$$ W_Q(\mathbf{x}, y) - \frac{1}{2} \;=\; \mathop{\mathbf{E}}_{h \sim Q}\left[ I(h(\mathbf{x}) \neq y) - \frac{1}{2} \right] \;=\; \mathop{\mathbf{E}}_{h \sim Q} -\frac{y h(\mathbf{x})}{2} \;=\; -\frac{1}{2}\sum_{h \in \mathcal{H}} Q(h) y h(\mathbf{x}) $$

$$ \;=\; -\frac{1}{2} y f_Q(\mathbf{x}) \,. $$

Since $\mathop{\mathbf{E}}_{(\mathbf{x}, y) \sim D} W_Q(\mathbf{x}, y)$ is the *Gibbs error rate* (by definition), we see that the expected margin is just one minus twice the Gibbs error rate. In contrast, the error for the $Q$-weighted majority vote is given by

$$ \mathop{\mathbf{E}}_{(\mathbf{x}, y) \sim D} I\left( W_Q(\mathbf{x}, y) > \frac{1}{2} \right) \;=\; \mathop{\mathbf{E}}_{(\mathbf{x}, y) \sim D} \lim_{\beta \to \infty} \frac{1}{2} \tanh\left( \beta \left[ 2 W_Q(\mathbf{x}, y) - 1 \right] \right) + \frac{1}{2} $$

$$ \leq \mathop{\mathbf{E}}_{(\mathbf{x}, y) \sim D} \tanh\left( \beta \left[ 2 W_Q(\mathbf{x}, y) - 1 \right] \right) + 1 \quad (\forall \beta > 0) $$

$$ \leq \mathop{\mathbf{E}}_{(\mathbf{x}, y) \sim D} \exp\left( \beta \left[ 2 W_Q(\mathbf{x}, y) - 1 \right] \right) \quad (\forall \beta > 0) \,. $$

Hence, for large enough $\beta$, the sigmoid loss (or $\tanh$ loss) of $f_Q$ should be very close to the error rate of the $Q$-weighted majority vote. Moreover, the error rate of the majority vote is always upper bounded by twice that sigmoid loss for any $\beta > 0$. The sigmoid loss is, in turn, upper bounded by the exponential loss (which is used, for example, in Adaboost [9]).

More generally, we will provide tight risk bounds for any loss function that can be expanded by a Taylor series around $W_Q(\mathbf{x}, y) = 1/2$. Hence we consider any loss function $\zeta_Q(\mathbf{x}, y)$ that can be written as

$$ \zeta_Q(\mathbf{x}, y) \;\overset{\text{def}}{=}\; \frac{1}{2} + \frac{1}{2}\sum_{k=1}^{\infty} g(k) \left( 2 W_Q(\mathbf{x}, y) - 1 \right)^k \tag{1} $$

$$ \;=\; \frac{1}{2} + \frac{1}{2}\sum_{k=1}^{\infty} g(k) \left( \mathop{\mathbf{E}}_{h \sim Q} - y h(\mathbf{x}) \right)^k \,, \tag{2} $$

and our task is to provide tight bounds for the expected loss $\zeta_Q$ that depend on the empirical loss $\widehat{\zeta_Q}$ measured on a training sequence $S = \langle (\mathbf{x}_1, y_1), \ldots, (\mathbf{x}_m, y_m) \rangle$ of $m$ examples, where

$$ \zeta_Q \;\overset{\text{def}}{=}\; \mathop{\mathbf{E}}_{(\mathbf{x}, y) \sim D} \zeta_Q(\mathbf{x}, y) \;\;;\;\; \widehat{\zeta_Q} \;\overset{\text{def}}{=}\; \frac{1}{m}\sum_{i=1}^{m} \zeta_Q(\mathbf{x}_i, y_i) \,. \tag{3} $$

Note that by upper bounding $\zeta_Q$, we are taking into account all moments of $W_Q$. In contrast, the PAC-Bayes theorem [2, 3, 4, 5] currently only upper bounds the first moment $\mathop{\mathbf{E}}_{(\mathbf{x}, y) \sim D} W_Q(\mathbf{x}, y)$.

# 3  A PAC-Bayes Risk Bound for Convex Combinations of Classifiers

The PAC-Bayes theorem [2, 3, 4, 5] is a statement about the expected zero-one loss of a Gibbs classifier. Given any distribution over a space of classifiers, the Gibbs classifier labels any example $\mathbf{x} \in \mathcal{X}$ according to a classifier randomly drawn from that distribution. Hence, to obtain a PAC-Bayesian bound for the expected general loss $\zeta_Q$ of a convex combination of classifiers, let us relate $\zeta_Q$ to the zero-one loss of a Gibbs classifier. For this task, let us first write

$$\mathop{\mathbf{E}}_{(\mathbf{x},y)\sim D} \left( \mathop{\mathbf{E}}_{h\sim Q} - yh(\mathbf{x}) \right)^k = \mathop{\mathbf{E}}_{h_1\sim Q} \mathop{\mathbf{E}}_{h_2\sim Q} \cdots \mathop{\mathbf{E}}_{h_k\sim Q} \mathop{\mathbf{E}}_{(\mathbf{x},y)} (-y)^k h_1(\mathbf{x})h_2(\mathbf{x})\cdots h_k(\mathbf{x}) .$$

Note that the product $h_1(\mathbf{x})h_2(\mathbf{x})\cdots h_k(\mathbf{x})$ defines another binary classifier that we denote as $h_{1-k}(\mathbf{x})$. We now define the error rate $R(h_{1-k})$ of $h_{1-k}$ as

$$R(h_{1-k}) \overset{\text{def}}{=} \mathop{\mathbf{E}}_{(\mathbf{x},y)\sim D} I\Big( (-y)^k h_{1-k}(\mathbf{x}) = \text{sgn}(g(k)) \Big) \tag{4}$$

$$= \frac{1}{2} + \frac{1}{2} \cdot \text{sgn}(g(k)) \mathop{\mathbf{E}}_{(\mathbf{x},y)\sim D} (-y)^k h_{1-k}(\mathbf{x}) ,$$

where $\text{sgn}(g) = +1$ if $g > 0$ and $-1$ otherwise.

If we now use $\mathop{\mathbf{E}}_{h_{1-k}\sim Q^k}$ to denote $\mathop{\mathbf{E}}_{h_1\sim Q} \mathop{\mathbf{E}}_{h_2\sim Q} \cdots \mathop{\mathbf{E}}_{h_k\sim Q}$, Equation 2 now becomes

$$\zeta_Q = \frac{1}{2} + \frac{1}{2}\sum_{k=1}^{\infty} g(k) \mathop{\mathbf{E}}_{(\mathbf{x},y)\sim D} \left( \mathop{\mathbf{E}}_{h\sim Q} - yh(\mathbf{x}) \right)^k$$

$$= \frac{1}{2} + \frac{1}{2}\sum_{k=1}^{\infty} |g(k)| \cdot \text{sgn}(g(k)) \mathop{\mathbf{E}}_{h_{1-k}\sim Q^k} \mathop{\mathbf{E}}_{(\mathbf{x},y)\sim D} (-y)^k h_{1-k}(\mathbf{x})$$

$$= \frac{1}{2} + \sum_{k=1}^{\infty} |g(k)| \mathop{\mathbf{E}}_{h_{1-k}\sim Q^k} \left( R(h_{1-k}) - \frac{1}{2} \right) . \tag{5}$$

Apart, from constant factors, Equation 5 relates $\zeta_Q$ the the zero-one loss of a new type of Gibbs classifier. Indeed, if we define

$$c \overset{\text{def}}{=} \sum_{k=1}^{\infty} |g(k)| , \tag{6}$$

Equation 5 can be rewritten as

$$\frac{1}{c}\left( \zeta_Q - \frac{1}{2} \right) + \frac{1}{2} = \frac{1}{c}\sum_{k=1}^{\infty} |g(k)| \mathop{\mathbf{E}}_{h_{1-k}\sim Q^k} R(h_{1-k}) \overset{\text{def}}{=} R(G_{\overline{Q}}) . \tag{7}$$

The new type of Gibbs classifier is denoted above by $G_{\overline{Q}}$, where $\overline{Q}$ is a distribution over the product classifiers $h_{1-k}$ with variable length $k$. More precisely, given an example $\mathbf{x}$ to be labelled by $G_{\overline{Q}}$, we first choose at random a number $k \in \mathbb{N}^+$ according to the discrete probability distribution given by $|g(k)|/c$ and then we choose $h_{1-k}$ randomly according to $Q^k$ to classify $\mathbf{x}$ with $h_{1-k}(\mathbf{x})$. The risk $R(G_{\overline{Q}})$ of this new Gibbs classifier is then given by Equation 7.

We will present a tight PAC-Bayesien bound for $R(G_{\overline{Q}})$ which will automatically translate into a bound for $\zeta_Q$ via Equation 7. This bound will depend on the empirical risk $R_S(G_{\overline{Q}})$ which relates to the the empirical loss $\widehat{\zeta_Q}$ (measured on the training sequence $S$ of $m$ examples) through the equation

$$\frac{1}{c}\left( \widehat{\zeta_Q} - \frac{1}{2} \right) + \frac{1}{2} = \frac{1}{c}\sum_{k=1}^{\infty} |g(k)| \mathop{\mathbf{E}}_{h_{1-k}\sim Q^k} R_S(h_{1-k}) \overset{\text{def}}{=} R_S(G_{\overline{Q}}) , \tag{8}$$

where

$$R_S(h_{1-k}) \overset{\text{def}}{=} \frac{1}{m}\sum_{i=1}^{m} I\Big( (-y_i)^k h_{1-k}(\mathbf{x}_i) = \text{sgn}(g(k)) \Big) .$$

Note that Equations 7 and 8 imply that

$$\zeta_Q - \widehat{\zeta_Q} \;=\; c \cdot \left[ R(G_{\overline{Q}}) - R_S(G_{\overline{Q}}) \right] \;.$$

Hence, any looseness in the bound for $R(G_Q)$ will be amplified by the scaling factor $c$ on the bound for $\zeta_Q$. Therefore, within this approach, the bound for $\zeta_Q$ can be tight only for small values of $c$. Note however that loss functions having a small value of $c$ are commonly used in practice. Indeed, learning algorithms for feed-forward neural networks, and other approaches that construct a real-valued function $f_Q(\mathbf{x}) \in [-1, +1]$ from binary classification data, typically use a loss function of the form $|f_Q(\mathbf{x}) - y|^r / 2$, for $r \in \{1, 2\}$. In these cases we have

$$\frac{1}{2} |f_Q(\mathbf{x}) - y|^r \;=\; \frac{1}{2} \left| \mathop{\mathbf{E}}_{h \sim Q} yh(\mathbf{x}) - 1 \right|^r \;=\; 2^{r-1} \left| W_Q(\mathbf{x}, y) \right|^r \;,$$

which gives $c = 1$ for $r = 1$, and $c = 3$ for $r = 2$.

Given a set $\mathcal{H}$ of classifiers, a prior distribution $P$ on $\mathcal{H}$, and a training sequence $S$ of $m$ examples, the learner will output a posterior distribution $Q$ on $\mathcal{H}$ which, in turn, gives a convex combination $f_Q$ that suffers the expected loss $\zeta_Q$. Although Equation 7 holds only for a distribution $\overline{Q}$ defined by the absolute values of the Taylor coefficients $g(k)$ and the product distribution $Q^k$, the PAC-Bayesian theorem will hold for any prior $\overline{P}$ and posterior $\overline{Q}$ defined on

$$\mathcal{H}^* \;\overset{\text{def}}{=}\; \bigcup_{k \in \mathbb{N}^+} \mathcal{H}^k \;, \tag{9}$$

and for any zero-one valued loss function $\ell(h(\mathbf{x}), y))$ defined $\forall h \in \mathcal{H}^*$ and $\forall (\mathbf{x}, y) \in \mathcal{X} \times \mathcal{Y}$ (not just the one defined by Equation 4). This PAC-Bayesian theorem upper-bounds the value of $\text{kl}\big(R_S(G_{\overline{Q}}) \big\| R(G_{\overline{Q}})\big)$, where

$$\text{kl}(q\|p) \;\overset{\text{def}}{=}\; q \ln \frac{q}{p} + (1 - q) \ln \frac{1 - q}{1 - p}$$

denotes the Kullback-Leibler divergence between the Bernoulli distributions with probability of success $q$ and probability of success $p$. Note that an upper bound on $\text{kl}\big(R_S(G_{\overline{Q}}) \big\| R(G_{\overline{Q}})\big)$ provides both and upper and a lower bound on $R(G_{\overline{Q}})$.

The upper bound on $\text{kl}\big(R_S(G_{\overline{Q}}) \big\| R(G_{\overline{Q}})\big)$ depends on the value of $\text{KL}(\overline{Q}\|\overline{P})$, where

$$\text{KL}(\overline{Q}\|\overline{P}) \;\overset{\text{def}}{=}\; \mathop{\mathbf{E}}_{h \sim \overline{Q}} \ln \frac{\overline{Q}(h)}{\overline{P}(h)}$$

denotes the Kullback-Leibler divergence between distributions $\overline{Q}$ and $\overline{P}$ defined on $\mathcal{H}^*$.

In our case, since we want a bound on $R(G_{\overline{Q}})$ that translates into a bound for $\zeta_Q$, we need a $\overline{Q}$ that satisfies Equation 7. To minimize the value of $\text{KL}(\overline{Q}\|\overline{P})$, it is desirable to choose a prior $\overline{P}$ having properties similar to those of $\overline{Q}$. Namely, the probabilities assigned by $\overline{P}$ to the possible values of $k$ will also be given by $|g(k)|/c$. Moreover, we will restrict ourselves to the case where the $k$ classifiers from $\mathcal{H}$ are chosen independently, each according to the prior $P$ on $\mathcal{H}$ (however, other choices for $\overline{P}$ are clearly possible). In this case we have

$$
\begin{aligned}
\text{KL}(\overline{Q}\|\overline{P}) &= \frac{1}{c} \sum_{k=1}^{\infty} |g(k)| \mathop{\mathbf{E}}_{h_{1-k} \sim Q^k} \ln \frac{|g(k)| \cdot Q^k(h_{1-k})}{|g(k)| \cdot P^k(h_{1-k})} \\
&= \frac{1}{c} \sum_{k=1}^{\infty} |g(k)| \mathop{\mathbf{E}}_{h_1 \sim Q} \cdots \mathop{\mathbf{E}}_{h_k \sim Q} \sum_{i=1}^{k} \ln \frac{Q(h_i)}{P(h_i)} \\
&= \frac{1}{c} \sum_{k=1}^{\infty} |g(k)| \cdot k \mathop{\mathbf{E}}_{h \sim Q} \ln \frac{Q(h)}{P(h)} \\
&= \overline{k} \cdot \text{KL}(Q\|P) \;, \tag{10}
\end{aligned}
$$

where

$$\overline{k} \stackrel{\text{def}}{=} \frac{1}{c} \sum_{k=1}^{\infty} |g(k)| \cdot k . \tag{11}$$

We then have the following theorem.

**Theorem 1** *For any set $\mathcal{H}$ of binary classifiers, any prior distribution $\overline{P}$ on $\mathcal{H}^*$, and any $\delta \in (0, 1]$, we have*

$$\Pr_{S \sim D^m} \left( \forall \overline{Q} \text{ on } \mathcal{H}^* : \text{kl}\big(R_S(G_{\overline{Q}}) \| R(G_{\overline{Q}})\big) \ \leq \ \frac{1}{m} \left[ \text{KL}(\overline{Q} \| \overline{P}) + \ln \frac{m+1}{\delta} \right] \right) \ \geq \ 1 - \delta .$$

**Proof** The proof directly follows from the fact that we can apply the PAC-Bayes theorem of [4] to priors and posteriors defined on the space $\mathcal{H}^*$ of binary classifiers with any zero-one valued loss function. ∎

Note that Theorem 1 directly provides upper and lower bounds on $\zeta_Q$ when we use Equations 7 and 8 to relate $R(G_{\overline{Q}})$ and $R_S(G_{\overline{Q}})$ to $\zeta_Q$ and $\widehat{\zeta_Q}$ and when we use Equation 10 for $\text{KL}(\overline{Q} \| \overline{P})$. Consequently, we have the following theorem.

**Theorem 2** *Consider any loss function $\zeta_Q(\mathbf{x}, y)$ defined by Equation 1. Let $\zeta_Q$ and $\widehat{\zeta_Q}$ be, respectively, the expected loss and its empirical estimate (on a sample of $m$ examples) as defined by Equation 3. Let $c$ and $\overline{k}$ be defined by Equations 6 and 11 respectively. Then for any set $\mathcal{H}$ of binary classifiers, any prior distribution $P$ on $\mathcal{H}$, and any $\delta \in (0, 1]$, we have*

$$\Pr_{S \sim D^m} \left( \forall Q \text{ on } \mathcal{H} : \text{kl}\left( \frac{1}{c}\left[\widehat{\zeta_Q} - \frac{1}{2}\right] + \frac{1}{2} \, \middle\| \, \frac{1}{c}\left[\zeta_Q - \frac{1}{2}\right] + \frac{1}{2} \right) \right.$$

$$\left. \leq \frac{1}{m} \left[ \overline{k} \cdot \text{KL}(Q \| P) + \ln \frac{m+1}{\delta} \right] \right) \ \geq \ 1 - \delta .$$

## 4 Bound Behavior During Adaboost

We have decided to examine the behavior of the proposed bounds during Adaboost since this learning algorithm generally produces a weighted majority vote having a large Gibbs risk $\mathbf{E}_{(\mathbf{x},y)} W_Q(\mathbf{x}, y)$ (*i.e.*, small expected margin) and a small $\mathbf{Var}_{(\mathbf{x},y)} W_Q(\mathbf{x}, y)$ (*i.e.*, small variance of the margin). Indeed, recall that one of our main motivations was to find a tight risk bound for the majority vote precisely under these circumstances.

We have used the "symmetric" version of Adaboost [10, 9] where, at each boosting round $t$, the weak learning algorithm produces a classifier $h_t$ with the smallest empirical error

$$\epsilon_t = \sum_{i=1}^{m} D_t(i) I[h_t(\mathbf{x}_i) \neq y_i]$$

with respect to the boosting distribution $D_t(i)$ on the indices $i \in \{1, \ldots, m\}$ of the training examples. After each boosting round $t$, this distribution is updated according to

$$D_{t+1}(i) = \frac{1}{Z_t} D_t(i) \exp(-y_i \alpha_t h_t(x_i)) ,$$

where $Z_t$ is the normalization constant required for $D_{t+1}$ to be a distribution, and where

$$\alpha_t = \frac{1}{2} \ln \left( \frac{1 - \epsilon_t}{\epsilon_t} \right) .$$

Since our task is not to obtain the majority vote with the smallest possible risk but to investigate the tightness of the proposed bounds, we have used the standard "decision stumps" for the set $\mathcal{H}$

of classifiers that can be chosen by the weak learner. Each decision stump is a threshold classifier that depends on a single attribute: it outputs $+y$ when the tested attribute exceeds the threshold and predicts $-y$ otherwise, where $y \in \{-1, +1\}$. For each decision stump $h \in \mathcal{H}$, its boolean complement is also in $\mathcal{H}$. Hence, we have $2[k(i) - 1]$ possible decision stumps on an attribute $i$ having $k(i)$ possible (discrete values). Hence, for data sets having $n$ attributes, we have exactly $|\mathcal{H}| = 2 \sum_{i=1}^{n} 2[k(i) - 1]$ classifiers. Data sets having continuous-valued attributes have been discretized in our numerical experiments.

From Theorem 2 and Equation 10, the bound on $\zeta_Q$ depends on $\text{KL}(Q\|P)$. We have chosen a uniform prior $P(h) = 1/|\mathcal{H}| \quad \forall h \in \mathcal{H}$. We therefore have

$$\text{KL}(Q\|P) = \sum_{h \in \mathcal{H}} Q(h) \ln \frac{Q(h)}{P(h)} = \sum_{h \in \mathcal{H}} Q(h) \ln Q(h) + \ln |\mathcal{H}| \stackrel{\text{def}}{=} -H(Q) + \ln |\mathcal{H}| .$$

At boosting round $t$, Adaboost changes the distribution from $D_t$ to $D_{t+1}$ by putting more weight on the examples that are incorrectly classified by $h_t$. This strategy is supported by the propose bound on $\zeta_Q$ since it has the effect of increasing the entropy $H(Q)$ as a function of $t$. Indeed, apart from tiny fluctuations, the entropy was seen to be nondecreasing as a function of $t$ in all of our boosting experiments.

We have focused our attention on two different loss functions: the exponential loss and the sigmoid loss.

## 4.1 Results for the Exponential Loss

The exponential loss $\mathcal{E}_Q(\mathbf{x}, y)$ is the obvious choice for boosting since, the typical analysis [8, 10, 9] shows that the empirical estimate of the exponential loss is decreasing at each boosting round [2]. More precisely, we have chosen

$$\mathcal{E}_Q(\mathbf{x}, y) \stackrel{\text{def}}{=} \frac{1}{2} \exp \left( \beta \left[ 2W_Q(\mathbf{x}, y) - 1 \right] \right) . \tag{12}$$

For this loss function, we have

$$c = e^{\beta} - 1$$
$$\overline{k} = \frac{\beta}{1 - e^{-\beta}} .$$

Since $c$ increases exponentially rapidly with $\beta$, so will the risk upper-bound for $\mathcal{E}_Q$. Hence, unfortunately, we can obtain a tight upper-bound only for small values of $\beta$.

All the data sets used were obtained from the UCI repository. Each data set was randomly split into two halves of the same size: one for the training set and the other for the testing set. Figure 1 illustrates the typical behavior for the exponential loss bound on the Mushroom and Sonar data sets containing 8124 examples and 208 examples respectively.

We first note that, although the test error of the majority vote (generally) decreases as function of the number $T$ of boosting rounds, the risk of the Gibbs classifier, $\mathbf{E}_{(\mathbf{x}, y)} W_Q(\mathbf{x}, y)$ *increases* as a function of $T$ but its variance $\mathbf{Var}_{(\mathbf{x}, y)} W_Q(\mathbf{x}, y)$ *decreases* dramatically. Another striking feature is the fact that the exponential loss bound curve, computed on the training set, is essentially parallel to the true exponential loss curve computed on the testing set. This same parallelism was observed for all the UCI data sets we have examined so far.[3] Unfortunately, as we can see in Figure 2, the risk bound increases rapidly as a function of $\beta$. Interestingly however, the risk bound curves remain parallel to the true risk curves.

## 4.2 Results for the Sigmoid Loss

We have also investigated the sigmoid loss $\mathcal{T}_Q(\mathbf{x}, y)$ defined by

$$\mathcal{T}_Q(\mathbf{x}, y) \stackrel{\text{def}}{=} \frac{1}{2} + \frac{1}{2} \tanh \left( \beta \left[ 2W_Q(\mathbf{x}, y) - 1 \right] \right) . \tag{13}$$

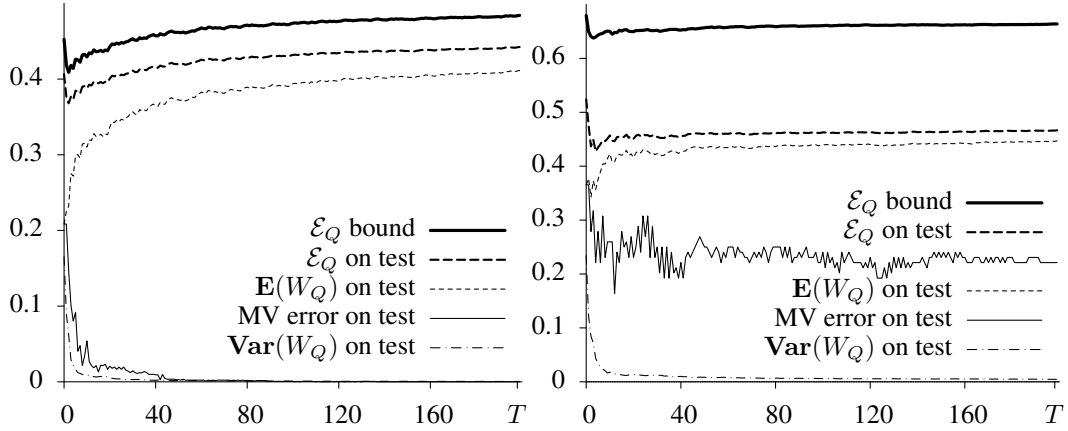

Figure 1: Behavior of the exponential risk bound ($\mathcal{E}_Q$ bound), the true exponential risk ($\mathcal{E}_Q$ on test), the Gibbs risk ($\mathbf{E}(W_Q)$ on test), its variance ($\mathbf{Var}(W_Q)$ on test), and the test error of the majority vote (MV error on test) as of function of the boosting round $T$ for the Mushroom (left) and the Sonar (right) data sets. The risk bound and the true risk were computed for $\beta = \ln 2$.

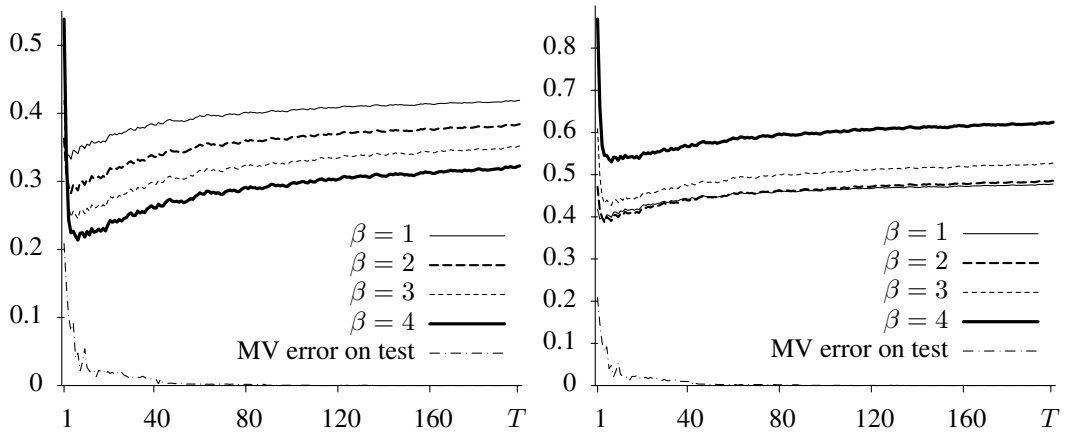

Figure 2: Behavior of the true exponential risk (left) and the exponential risk bound (right) for different values of $\beta$ on the Mushroom data set.

Since the Taylor series expansion for $\tanh(x)$ about $x = 0$ converges only for $|x| < \pi/2$, we are limited to $\beta \leq \pi/2$. Under these circumstances, we have

$$
\begin{aligned}
c &= \tan(\beta) \\
\overline{k} &= \frac{1}{\cos(\beta)\sin(\beta)} \ .
\end{aligned}
$$

Similarly as in Figure 1, we see on Figure 3 that the sigmoid loss bound curve, computed on the training set, is essentially parallel to the true sigmoid loss curve computed on the testing set. Moreover, the bound appears to be as tight as the one for the exponential risk on Figure 1.

## 5  Conclusion

By trying to obtain a tight PAC-Bayesian risk bound for the majority vote, we have obtained a PAC-Bayesian risk bound for any loss function $\zeta_Q$ that has a convergent Taylor expansion around $W_Q = 1/2$ (such as the exponential loss and the sigmoid loss). Unfortunately, the proposed risk

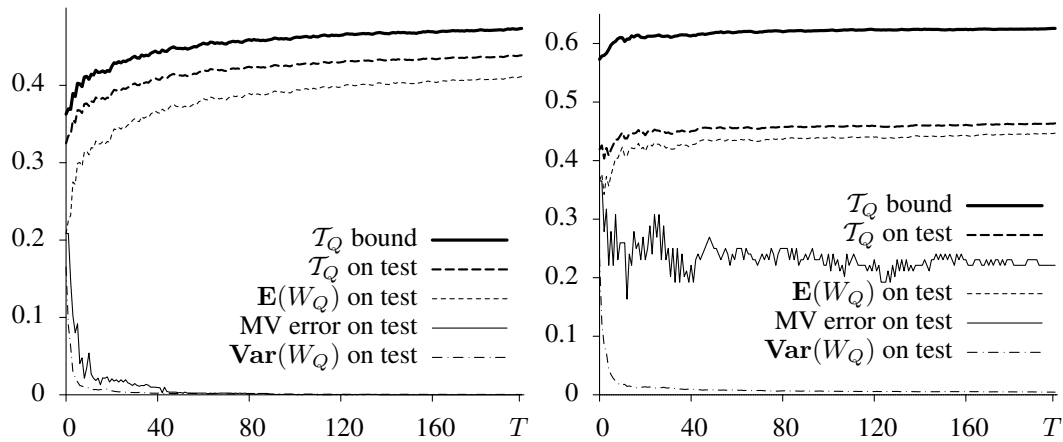

Figure 3: Behavior of the sigmoid risk bound ($\mathcal{T}_Q$ bound), the true sigmoid risk ($\mathcal{T}_Q$ on test), the Gibbs risk ($\mathbf{E}(W_Q)$ on test), its variance ($\mathbf{Var}(W_Q)$ on test), and the test error of the majority vote (MV error on test) as of function of the boosting round $T$ for the Mushroom (left) and the Sonar (right) data sets. The risk bound and the true risk were computed for $\beta = \ln 2$.

bound is tight only for small values of the scaling factor $c$ involved in the relation between the expected loss $\zeta_Q$ of a convex combination of binary classifiers and the zero-one loss of a related Gibbs classifier $G_{\overline{Q}}$. However, it is quite encouraging to notice in our numerical experiments with Adaboost that the proposed loss bound (for the exponential loss and the sigmoid loss), behaves very similarly as the true loss.

## Acknowledgments

Work supported by NSERC Discovery grants 262067 and 122405.

## Footnotes

[1] When $\mathcal{H}$ is a continuous set, $Q(h)$ denotes a density and the summations over $h$ are replaced by integrals.

[2] In fact, this is true only for the *positive linear combination* produced by Adaboost. The empirical exponential risk of the *convex combination* $f_Q$ is not always decreasing as we shall see.

[3] These include the following data sets: Wisconsin-breast, breast cancer, German credit, ionosphere, kr-vs-kp, USvotes, mushroom, and sonar.

## References

[1] David McAllester. Some PAC-Bayesian theorems. *Machine Learning*, 37:355–363, 1999.

[2] Matthias Seeger. PAC-Bayesian generalization bounds for gaussian processes. *Journal of Machine Learning Research*, 3:233–269, 2002.

[3] David McAllester. PAC-Bayesian stochastic model selection. *Machine Learning*, 51:5–21, 2003.

[4] John Langford. Tutorial on practical prediction theory for classification. *Journal of Machine Learning Research*, 6:273–306, 2005.

[5] François Laviolette and Mario Marchand. PAC-Bayes risk bounds for sample-compressed Gibbs classifiers. *Proceedings of the 22nth International Conference on Machine Learning (ICML 2005)*, pages 481–488, 2005.

[6] John Langford and John Shawe-Taylor. PAC-Bayes & margins. In S. Thrun S. Becker and K. Obermayer, editors, *Advances in Neural Information Processing Systems 15*, pages 423–430. MIT Press, Cambridge, MA, 2003.

[7] Leo Breiman. Bagging predictors. *Machine Learning*, 24:123–140, 1996.

[8] Yoav Freund and Robert E. Schapire. A decision-theoretic generalization of on-line learning and an application to boosting. *Journal of Computer and System Sciences*, 55:119–139, 1997.

[9] Robert E. Schapire and Yoram Singer. Improved bosting algorithms using confidence-rated predictions. *Machine Learning*, 37:297–336, 1999.

[10] Robert E. Schapire, Yoav Freund, Peter Bartlett, and Wee Sun Lee. Boosting the margin: A new explanation for the effectiveness of voting methods. *The Annals of Statistics*, 26:1651–1686, 1998.
